# ProtGO: Function-Guided Protein Modeling for Unified Representation Learning

**Bozhen Hu**[1,2*]**, Cheng Tan**[2*]**, Yongjie Xu**[2]**, Zhangyang Gao**[2]**, Jun Xia**[2]**,**
**Lirong Wu**[2]**, Stan Z. Li**[2†]
[1]Zhejiang University    [2]Westlake University
{hubozhen, tancheng, stan.zq.li}@westlake.edu.cn

## Abstract

Protein representation learning is indispensable for various downstream applications of artificial intelligence for bio-medicine research, such as drug design and function prediction. However, achieving effective representation learning for proteins poses challenges due to the diversity of data modalities involved, including sequence, structure, and function annotations. Despite the impressive capabilities of large language models in biomedical text modelling, there remains a pressing need for a framework that seamlessly integrates these diverse modalities, particularly focusing on the three critical aspects of protein information: sequence, structure, and function. Moreover, addressing the inherent data scale differences among these modalities is essential. To tackle these challenges, we introduce ProtGO, a unified model that harnesses a teacher network equipped with a customized graph neural network (GNN) and a Gene Ontology (GO) encoder to learn hybrid embeddings. Notably, our approach eliminates the need for additional functions as input for the student network, which shares the same GNN module. Importantly, we utilize a domain adaptation method to facilitate distribution approximation for guiding the training of the teacher-student framework. This approach leverages distributions learned from latent representations to avoid the alignment of individual samples. Benchmark experiments highlight that ProtGO significantly outperforms state-of-the-art baselines, clearly demonstrating the advantages of the proposed unified framework.

## 1 Introduction

Proteins constitute the fundamental structural and functional components within cells and organisms and serve as indispensable biomolecules thereof. These biomolecules are composed of linear sequences of amino acids, linked together by peptide bonds, intricately folding into complex three-dimensional (3D) structures [1]. Recent groundbreaking advancements, exemplified by AlphaFold models [2, 3], have revolutionized protein structure prediction, leveraging artificial intelligence techniques with unprecedented accuracy. Consequently, a significant scientific challenge emerges: unravelling the intricate relationships among a protein's sequence, structure, and function — an endeavour crucial for understanding disease mechanisms [4]. Protein science primarily encompasses three core types of information: sequence, structure, and function, as illustrated in Figure 1. There, Gene Ontology (GO) annotations offer a standardized framework for delineating gene and protein functions, covering molecular functions, cellular components, and biological processes. These annotations furnish a comprehensive grasp of protein functionality across multiple dimensions and

are the purpose of a protein study. Designed to be species-agnostic, the GO vocabulary spans a diverse array of biological functions, rendering it applicable to various biological contexts and organisms[1] .

Protein representation learning emerges as a dynamic research domain, aiming to uncover underlying patterns within raw protein data, providing invaluable insights applicable across diverse downstream tasks [5]. Recently, protein language models (LMs) have emerged as powerful tools for processing protein sequences, showcasing their capability to learn the certain 'grammar of life' from vast collections of protein sequences [6]. Models such as ProtTrans [7], the ESM series [6, 8–10], and xTrimoPGLM [11] leverage transformer architectures and attention mechanisms to autonomously uncover intrinsic patterns, undergoing self-supervised pre-training on extensive datasets. In contrast to sequences, protein structures entail continuous 3D coordinate data [12], necessitating distinct modelling strategies. Graph neural network (GNN)-based models [13, 14] have been developed and adapted to represent protein 3D structures [15, 16]. For instance, GearNet [17] encodes both sequential and spatial features by facilitating message passing between nodes and edges across multiple types of protein graphs.

While protein LMs and GNNs have demonstrated impressive performance across various protein-related tasks, such as predicting protein mutational stability and Enzyme Commission (EC) numbers [18], most of these methods overlook the utilization of functional information [6, 7]. However, integrating functional annotations is crucial for enhancing model capabilities and unveiling the intrinsic relationships between protein sequences and functions [18, 19]. Recent research explores token-level protein knowledge through pre-training on biomedical texts, which contain sequential and biological information [19, 20]. Nonetheless, the disparity between the vast number of protein sequences and the limited availability of

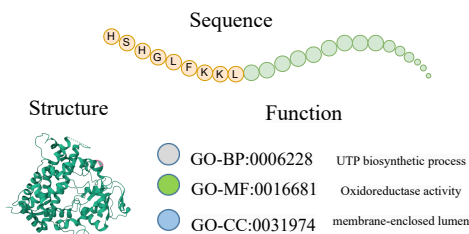

Figure 1: Protein sequence, structure, and function.

structures and annotations presents a significant challenge [21]. For instance, UniParc contains over 500 million (M) sequences [22], whereas the Protein Data Bank (PDB) houses only about 190 thousand (K) structures [23], with approximately 5M triplets in ProteinKG25 [24], comprising around 600K proteins and 50K attribute terms. This discrepancy in scale hampers the translation of the success achieved in sequence modelling into structure and function modelling. Consequently, enhancing the integration and fusion of information from these three modalities poses a crucial and formidable challenge in protein science.

Considering the disparity in data categories and scales between protein sequences, structures, and functions, we introduce ProtGO, a comprehensive multimodal framework for protein representation learning. ProtGO leverages a teacher model to glean insights from triplets comprising sequences, structures, and functions, distilling this knowledge to guide the training of the student model. However, given that functional annotations are lacking for the vast majority of sequenced proteins [25, 26], such information may not always be available for the teacher model in downstream tasks. Consequently, we opt to train a sequence-structure student model, which can be readily applied to various downstream tasks, with the teacher model serving solely to provide functional knowledge to the student model. To facilitate knowledge transfer from teacher to student, we employ domain adaptation techniques to align the distributions of latent spaces between the teacher and student models. Specifically, we minimize the Kullback-Leibler (KL) divergence to mitigate distribution discrepancies between the teacher and student domains.

The contributions of this paper can be summarized: **1)** We propose ProtGO to integrate multimodal information about proteins, encompassing sequence, structure, and functions. The teacher-student framework enables the learning of unified representations suitable for diverse downstream tasks where a protein model is needed. **2)** We pioneer the adaptation of knowledge distillation methods to connect protein teacher-student networks, infusing functional information into student representations through distribution approximation and domain adaptation. **3)** We validate ProtGO by outperforming prevailing protein representation methods across a range of tasks, including protein fold prediction, enzyme reaction classification, GO term prediction, and EC number prediction.

## 2 Related Work

### 2.1 Representation Learning for Protein

**Unimodal Protein Representation Learning.** There are two primary categories of methods in unimodal protein representation learning: sequence-based and structure-based approaches. Sequence-based methods aim to derive representations directly from amino acid sequences [27], with notable efforts focused on enhancing model sizes or scaling datasets [7, 10, 27–30]. For instance, Chen et al. proposed xTrimoPGLM, a unified model capable of learning from protein sequences to address both protein understanding and generation tasks concurrently, boasting a staggering 100 billion (B) parameters and 1 trillion (T) training tokens. In addition to sequence-based approaches, structure-based encoders have emerged to leverage the 3D structural information of proteins. For example, IEConv [31] accommodates the inherent inductive bias in protein structure modelling by introducing a graph convolution layer that integrates intrinsic and extrinsic distances between nodes. ProNet [32] provides geometric representations across multiple levels of structure granularity.

**Multimodal Protein Representation Learning.** In the realm of protein multimodality learning, methodologies such as GearNet [17] have been devised to concurrently exploit both sequence and structural information. GearNet represents sequential and geometric features as graph node and edge features, employing a message-passing mechanism to encode them [31, 33]. CDConv [12] introduces continuous-discrete convolution to model sequential and geometric features. Acknowledging the SE(3)-equivariant properties in protein structures, equivariant and invariant features are tailored as model inputs [33, 34]. Furthermore, the integration of factual biological knowledge has demonstrated enhancements in pre-trained LMs on protein sequences [24]. KeAP [19] and ProtST [20] train biomedical LMs using masked language modelling [35] as the pretext task. Particularly noteworthy, MASSA [18] initially derives sequence-structure embeddings from existing pre-trained models [10, 33], subsequently aligning them globally with GO embeddings using five pre-training objectives.

### 2.2 Knowledge Distillation

Knowledge distillation involves the process of transferring knowledge from a larger teacher model to a smaller student model [36, 37]. Significant advancements have been made in graph-based knowledge distillation, leading to the development of various methodologies [38, 39]. RDD [40] mandates the student model to faithfully replicate the complete node embeddings of the teacher, ensuring the transfer of more informative knowledge. Another notable approach, GraphAKD [41], employs adversarial learning to distill node representations from the teacher to the student. This method effectively distills knowledge from both local and global perspectives, exhibiting superior performance compared to earlier graph distillation techniques [42].

### 2.3 Domain Adaptation

Domain adaptation aims to develop a model from labelled data in a source domain that can be effectively applied to a target domain by minimizing dissimilarities between their distributions [43–45]. Methods for distribution alignment focus on reducing disparities in both marginal and conditional representation distributions between the source and target domains [46, 47]. Adversarial learning techniques have demonstrated remarkable effectiveness in mitigating the discrepancy between source and target domains [48–50]. Semi-supervised domain adaptation strategies aim to minimize the source-target gap with limited labelled target data [51–54]. In our work, we apply domain adaptation methods to align the distributions of representations learned by teacher and student networks to avoid dependence on individual samples.

## 3 Method

### 3.1 Preliminaries

Here, we provide definitions and relevant notations for the problem, and background knowledge of the local coordinate system (LCS).

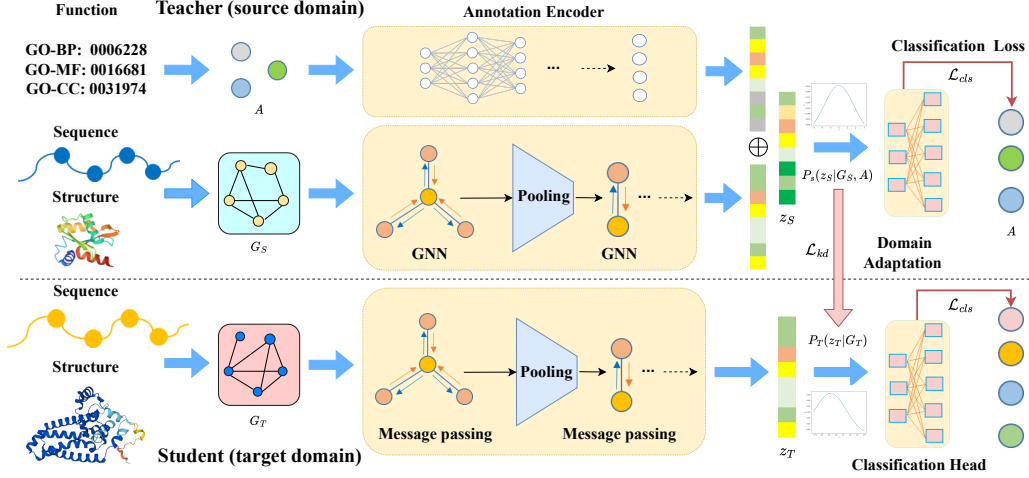

Figure 2: The overall framework of ProtGO consists of two branches: a teacher model in the source domain and a student model in the target domain, connected by a knowledge distillation loss.

**Problem Statement.** Mathematically, we represent a protein graph as $G = (\mathcal{V}, \mathcal{E}, X, E)$, where $\mathcal{V} = \{v_i\}_{i=1,\ldots,n}$ and $\mathcal{E} = \{\varepsilon_{ij}\}_{i,j=1,\ldots,n}$ denote the vertex and edge sets of $n$ residues, respectively. We denote the position of a residue by the coordinate of $C_\alpha$, the collection by the position matrix is denoted as $\mathcal{P} = \{P_i\}_{i=1,\ldots,n}$, where $P_i \in \mathbb{R}^{3\times1}$. The node and edge feature matrices are $X = [\boldsymbol{x}_i]_{i=1,\ldots,n}$ and $E = [\boldsymbol{e}_{ij}]_{i,j=1,\ldots,n}$, the feature vectors of node and edge are $\boldsymbol{x}_i \in \mathbb{R}^{d_1}$ and $\boldsymbol{e}_{ij} \in \mathbb{R}^{d_2}$, $d_1$ and $d_2$ are the initial feature dimensions. The set of $k$ GO annotations for proteins is denoted as $A = \{A_i\}_{i=1,\ldots,k}$, where $A_i \in \{0,1\}$ is the indicator for annotation $i$. The goal of protein graph representation learning is to find a low-dimensional embedding $z$ for each protein.

There is a source domain $S$ for the teacher model with the data distribution $p_S(z_S|G_S, A)$ in the latent space, and there is also a target domain $T$ for the student model with the data distribution $p_T(z_T|G_T)$ in the latent space. $z_S, z_T$ are latent embeddings from the teacher and student networks for protein graphs $G_S$ and $G_T$.

**Local Coordinate System.** In order to avoid the usage of complicated SE(3)-equivariant models, the invariant and locally informative features are developed from the LCS [55], which is defined as:

$$\boldsymbol{O}_i = [\boldsymbol{b_i} \quad \boldsymbol{n_i} \quad \boldsymbol{b_i} \times \boldsymbol{n_i}] \tag{1}$$

where $\boldsymbol{u}_i = \frac{P_i - P_{i-1}}{\|P_i - P_{i-1}\|}, \boldsymbol{b}_i = \frac{\boldsymbol{u}_i - \boldsymbol{u}_{i+1}}{\|\boldsymbol{u}_i - \boldsymbol{u}_{i+1}\|}, \boldsymbol{n}_i = \frac{\boldsymbol{u}_i \times \boldsymbol{u}_{i+1}}{\|\boldsymbol{u}_i \times \boldsymbol{u}_{i+1}\|}$, $b_i$ is the negative bisector of the angle between the rays $(P_{i-1} - P_i)$ and $(P_{i+1} - P_i)$.

$$\boldsymbol{e}_{ij} = \text{Concat}(\|P_i - P_j\|, \boldsymbol{O}_i^T \cdot \frac{P_i - P_j}{\|P_i - P_j\|}, \boldsymbol{O}_i^T \cdot \boldsymbol{O}_j) \tag{2}$$

Note that the edge feature vector $\boldsymbol{e}_{ij}$ is the concatenation of the geometric features for protein 3D structures, including distance, direction, and orientation, where $\|\cdot\|$ denotes the $l^2$-norm.

## 3.2 Overall Framework

The overall framework of ProtGO is illustrated in Figure 2. It consists of two branches that train a teacher model and a student model via iterative knowledge distillation. Compared to the student, the teacher has an additional annotation encoder module comprised of several fully connected layers. This transforms GO annotations into functional embeddings, combined with sequence-structure embeddings from the GNNs to form the final knowledge-enhanced embeddings $z_S$. Previous works have successfully utilized label-augmented techniques to enhance model training [56, 57]. This technique involves encoding labels and combining them with node attributes through concatenation or summation. By doing so, it improves feature representation and enables the model to effectively utilize valuable information from labels. Importantly, instead of directly minimizing the distances

between sample-dependent embeddings, denoted as $z_S$ and $z_T$, we introduce a sample-independent method. This is accomplished by aligning the latent space of the student with that of the teacher, achieved through the approximation of distributions of embeddings from both networks. This distribution alignment approach avoids dependence on individual sample inputs. It is noteworthy that our primary objective is to derive comprehensive embeddings for the student model, with less emphasis on the training specifics of the teacher model. Consequently, the teacher model can be trained on multiple datasets, whereas the student does not need to have access to the same datasets when data modalities are not available.

**Protein Graph Message Passing.** A protein sequence consists of $n$ residues, which are deemed as graph nodes. We concatenate the one-hot encoding of residue types with the physicochemical properties of each residue, namely, a steric parameter, hydrophobicity, volume, polarizability, isoelectric point, helix probability, and sheet probability [58, 59], which are used as the graph node features $\boldsymbol{x}_i$. These node features capture meaningful biochemical characteristics, enabling the model to learn which residues tend to be buried, exposed, tightly packed, etc. We define the sequential distance, $l_{ij} = \|i - j\|$, and spatial distance $d_{ij} = \|P_i - P_j\|$, where $P_i$ is the 3D coordinate of the $C_\alpha$ atom of the $i$-th residue. An edge $\varepsilon_{ij}$ exists if:

$$l_{ij} < l_s \quad \text{and} \quad d_{ij} < r_s \tag{3}$$

where $l_s, r_s$ are predefined radius thresholds, $\boldsymbol{e}_{ij}$ consists of geometric features of the protein structure, defined in Eq. 2. We convolve node and edge features from sequence and structure simultaneously and formulate the message passing mechanism as:

$$
\begin{aligned}
\boldsymbol{h}_i^{(0)} &= \mathrm{BN}\left(\mathrm{FC}\left(\boldsymbol{x}_i\right)\right), \\
\boldsymbol{u}_i^{(l)} &= \sigma(\mathrm{BN}(\sum_{v_j \in \mathcal{N}(v_i)} W\boldsymbol{e}_{ij}\boldsymbol{h}_j^{(l-1)})), \\
\boldsymbol{h}_i^{(l)} &= \boldsymbol{h}_i^{(l)} + \mathrm{Dropout}(\mathrm{FC}(\boldsymbol{u}_i^{(l)}))
\end{aligned}
\tag{4}
$$

This mechanism (as shown in Eq. 4) can fuse and update the node and edge features, which includes aggregation and update functions, where $\mathrm{FC}(\cdot)$, $\mathrm{BN}(\cdot)$, $\mathrm{Dropout}(\cdot)$ represent fully connected, batch normalization, and dropout layers, $\sigma(\cdot)$ is the activation function LeakyReLU and $W$ is the learnable convolutional kernel. $\mathcal{N}(v_i)$ refers to the neighbors of node $v_i$, and $\boldsymbol{h}_i^{(l)}$ is the representation of node $v_i$ in the $l$-th message passing layer. The node and edge features are processed together in Eq. 4. After message passing operations, a sequence pooling layer is applied to reduce the sequence length, providing a simple but effective way to aggregate key patterns. After average pooling, the residue number is halved; we expand the radius $r_s$ to $2r_s$ to update the edge conditions and perform the message passing and pooling operations again. These operations can make the GNNs cover more distant nodes gradually. The teacher and student models share the same GNN architecture to process protein sequences and structures. Finally, a global pooling layer is applied to obtain the graph-level protein embeddings, denoted as $h_S$ and $z_T$ for the teacher and student. Detailed model descriptions are presented in Appendix E.1.

**Protein Domain Adaption.** As shown in Figure 2, the teacher model consists of GNNs and an auxiliary annotation encoder, which is a multi-layer perceptron (MLP) that provides function-friendly protein representations. The annotations associated with $G_S$ serve as the input for the annotation encoder, resulting in the extraction of feature vector $h_A$. Therefore, we can combine $h_A$ and the graph-level protein embeddings $h_S$ learned from $G_S$ together:

$$
\begin{aligned}
h_A &= \mathrm{MLP}(A) \\
z_S &= h_A + \alpha h_S
\end{aligned}
\tag{5}
$$

where $\alpha$ is a hyper-parameter, controlling the balance between the contribution of the annotation embeddings $h_A$ and the protein embeddings $h_S$ in the combined representations.

As depicted in Figure 2, the generated protein embeddings $z_S$ contain sequence, structure, and function information, guiding the training of the student model. Since knowledge-enhanced embeddings $z_S$ are intended to be aligned with $z_T$, we obtain $z_S$ from the entire protein and GO term datasets to avoid dependence on individuals. Then, we calculate the distributions of $z_S$ and $z_T$ to better capture

the inherent uncertainty in the teacher's and student's latent spaces, in which the real distributions are sample-independent. The minibatch is adopted to approximate the quantities $p_S(z_S)$ and $p_T(z_T)$:

$$
\begin{aligned}
p_S(z_S) &= \mathbb{E}_{p_S(G_S,A)}[p(z_S|G_S,A)] \\
&\approx \frac{1}{B_S} \sum_{i=1}^{B_S} p_S(z_S|G_S^{(i)}, A^{(i)}) \\
p_T(z_T) &= \mathbb{E}_{p_T(G_T)}[p_T(z_T|G_T)] \\
&\approx \frac{1}{B_S} \sum_{i=1}^{B_S} p_T(z_T|G_T^{(i)})
\end{aligned}
\tag{6}
$$

where $B_S$ is the batch size. A Gaussian distribution $\Theta$ is assumed for protein embeddings, which exhibit smoothness and symmetry properties that can reasonably mimic the expected continuity and unimodality of the embeddings aggregated over many residues. We employ the reparameterization trick [60] to sample the protein embeddings.

$$
p_S(z_S) = \Theta(\mu_S, \sigma_S^2); \quad p_T(z_T) = \Theta(\mu_T, \sigma_T^2)
\tag{7}
$$

where $\mu_S, \sigma_S^2$ and $\mu_T, \sigma_T^2$ are the mean and variance values of the embeddings for the teacher and student models, providing a summary of the distribution using first- and second-order statistics.

Proposition 2 in Appendix F shows that the conditional misalignment in the representation space is bounded by the conditional misalignment in the input space. We have:

$$
\mathcal{L}_{\text{student}}^* \leq \mathcal{L}_{\text{teacher}} + \frac{M}{\sqrt{2}} C
\tag{8}
$$

$$
C = \sqrt{\text{KL}\left[p_S(z) \parallel p_T(z)\right] + \mathbb{E}_{p_S(G)}\left[\text{KL}\left[p_S(y|G) \parallel p_T(y|G)\right]\right]}
\tag{9}
$$

where $\mathcal{L}_{\text{student}}^*$ is the ideal target domain loss, and $\mathcal{L}_{\text{teacher}}$ is the teacher's supervised loss, $M$ is a bound, see Appendix F. $\mathbb{E}_{p_S(G)}\left[\text{KL}\left[p_S(y|G) \parallel p_T(y|G)\right]\right]$ is often small and fixed (not dependent on the representation $z$, and $y$ is the function label). To reduce the generalization bound, we can focus on optimizing the marginal misalignment with a hyper-parameter $\beta$:

$$
\mathcal{L}_{\text{teacher}} + \beta(\text{KL}\left[p_S(z) \parallel p_T(z)\right])
\tag{10}
$$

Eq. 10 can be used in an unsupervised way for the student to predict functions, which is near the ideal target domain loss. For the proposed framework, ProtGO (shown in Figure 2), we use the $\mathcal{L}_{\text{teacher}}$ to train the teacher model. When the student model has task labels, we adopt a hybrid loss $\mathcal{L}$ to train the student model, where the $\mathcal{L}_{kd} = \text{KL}\left[p_S(z)|p_T(z)\right]$ is to optimize the marginal misalignment between teacher and student models. Therefore, the final loss $\mathcal{L}$ with a hyper-parameter $\beta$ for the student model is formulated as:

$$
\mathcal{L} = \mathcal{L}_{\text{student}} + \beta \mathcal{L}_{kd}
\tag{11}
$$

The objective function of the teacher model $\mathcal{L}_{\text{teacher}}$ is the cross entropy loss for protein graph classification. The hybrid loss for the student model has a cross entropy loss $\mathcal{L}_{\text{student}}$ for classification and a regularization loss $\mathcal{L}_{kd}$ for domain-adapted knowledge distillation.

## 4 Experiments

### 4.1 Training Details

The proposed multimodal knowledge distillation framework, ProtGO, undergoes an easy training process. A dataset comprising approximately 30,000 proteins, each associated with 2,752 GO annotations from the GO dataset, is utilized without further categorization into biological process (BP), molecular function (MF), and cellular component (CC) classes [67]. These classes serve as input to the annotation encoder of the teacher model, yielding an overall $F_{\max}$ of 0.489 for the teacher model. Subsequently, the student model is trained. The optimization is performed using the Adam optimizer through the PyTorch library, and the performance metrics are computed as mean values over three initializations. Further details regarding experimental settings are available in Appendix E.2.

Table 1: Accuracy (%) of fold classification and enzyme reaction classification. The best results are shown in bold.

| Modality (Input) | Method | Fold Classification | | | Enzyme |
|---|---|---|---|---|---|
| | | Fold | SuperFamily | Family | Reaction |
| Sequence | CNN [61] | 11.3 | 13.4 | 53.4 | 51.7 |
| | ResNet [27] | 10.1 | 7.21 | 23.5 | 24.1 |
| | LSTM [27] | 6.41 | 4.33 | 18.1 | 11.0 |
| | Transformer [27] | 9.22 | 8.81 | 40.4 | 26.6 |
| Structure | GCN [62] | 16.8 | 21.3 | 82.8 | 67.3 |
| | GAT [63] | 12.4 | 16.5 | 72.7 | 55.6 |
| | 3DCNN_MQA [64] | 31.6 | 45.4 | 92.5 | 72.2 |
| Sequence-Structure | GraphQA [65] | 23.7 | 32.5 | 84.4 | 60.8 |
| | GVP [66] | 16.0 | 22.5 | 83.8 | 65.5 |
| | ProNet-Amino Acid [32] | 51.5 | 69.9 | 99.0 | 86.0 |
| | ProNet-Backbone [32] | 52.7 | 70.3 | 99.3 | 86.4 |
| | ProNet-All-Atom [32] | 52.1 | 69.0 | 99.0 | 85.6 |
| | CRL [16] | 47.6 | 70.2 | 99.2 | 87.2 |
| | GearNet [17] | 28.4 | 42.6 | 95.3 | 79.4 |
| | GearNet-IEConv [17] | 42.3 | 64.1 | 99.1 | 83.7 |
| | GearNet-Edge [17] | 44.0 | 66.7 | 99.1 | 86.6 |
| | GearNet-Edge-IEConv [17] | 48.3 | 70.3 | 99.5 | 85.3 |
| | CDConv [12] | 56.7 | 77.7 | 99.6 | 88.5 |
| | ProtGO (Student) | **60.5** | **79.4** | **99.8** | **89.4** |

## 4.2 Tasks and Baselines

Following the tasks in IEconv [31] and CDConv [12], we evaluate ProtGO on four protein tasks: protein fold classification, enzyme reaction classification, GO term prediction, and EC number prediction. Detailed task descriptions are presented in Appendix B. Dataset statistics are shown in Table 4 in the appendix.

**Baselines.** The proposed method is compared with existing protein representation learning methods, which are classified into three categories based on their inputs, which could be a sequence, 3D structure, or both sequence and structure. 1) Sequence-based encoders, including CNN [61], ResNet [27], LSTM [27] and Transformer [27]. 2) Structure-based methods (GCN [62], GAT [63], 3DCNN_MQA [64] 3) Sequence-structure based models, e.g., GVP [66], CRL [16], ProNet [32], GearNet [17], CDConv [12], etc. GearNet-IEConv and GearNetEdge-IEConv [17] add the IEConv [31] layer on GearNet.

## 4.3 Results of Fold and Enzyme Reaction Classification

Table 1 presents a performance comparison for protein fold and enzyme reaction prediction across various methods, with results reported as average values. As depicted in the table, the proposed ProtGO consistently achieves the highest performance across all four test sets for both fold and reaction prediction tasks. The superiority of sequence-structure-based methods over sequence- or structure-only approaches is evident, underscoring the advantages of jointly modelling sequence and structure information. ProtGO achieves a remarkable improvement in accuracy of over $6.7\%$ compared to prior techniques on the Fold test set, highlighting its efficacy in learning mappings between sequence, structure, and function. Furthermore, despite both CDConv and ProtGO employing sequence-structure convolution architectures, ProtGO demonstrates superior performance over the CDConv model. This observation suggests that the teacher-student training paradigm adopted in ProtGO facilitates the acquisition of enhanced protein embeddings by the student model.

Table 2: $F_{max}$ of GO term prediction and EC number prediction. The best results are shown in bold.

| Modality (Input) | Method | GO-BP | GO-MF | GO-CC | EC |
|---|---|---|---|---|---|
| Sequence | CNN [61] | 0.244 | 0.354 | 0.287 | 0.545 |
| | ResNet [27] | 0.280 | 0.405 | 0.304 | 0.605 |
| | LSTM [27] | 0.225 | 0.321 | 0.283 | 0.425 |
| | Transformer [27] | 0.264 | 0.211 | 0.405 | 0.238 |
| Structure | GCN [62] | 0.252 | 0.195 | 0.329 | 0.320 |
| | GAT [63] | 0.284 | 0.317 | 0.385 | 0.368 |
| | 3DCNN_MQA [64] | 0.240 | 0.147 | 0.305 | 0.077 |
| Sequence-Structure | GraphQA [65] | 0.308 | 0.329 | 0.413 | 0.509 |
| | GVP [66] | 0.326 | 0.426 | 0.420 | 0.489 |
| | CRL [16] | 0.421 | 0.624 | 0.431 | - |
| | GearNet [17] | 0.356 | 0.503 | 0.414 | 0.730 |
| | GearNet-IEConv [17] | 0.381 | 0.563 | 0.422 | 0.800 |
| | GearNet-Edge [17] | 0.403 | 0.580 | 0.450 | 0.810 |
| | GearNet-Edge-IEConv [17] | 0.400 | 0.581 | 0.430 | 0.810 |
| | CDConv [12] | 0.453 | 0.654 | 0.479 | 0.820 |
| | ProtGO (Student) | **0.464** | **0.667** | **0.492** | **0.857** |

## 4.4 Results of GO Term and EC Number Prediction

Following the protocol in GearNet [17], the test sets for GO term and EC number prediction only contain PDB chains with less than $95\%$ sequence identity to the training set, ensuring rigorous evaluation. The student model conducts the experiments, and the teacher model's annotations are not classified into these classes, avoiding data leakage. Table 2 shows comparative results between different protein modeling methods on these tasks, with performance measured by $F_{max}$, which balances precision and recall, working well even if positive and negative classes are imbalanced. The mean values of three independent runs are reported. ProtGO achieves the highest $F_{max}$ across all test sets for both GO and EC prediction, outperforming other approaches. This demonstrates ProtGO's strong capabilities for predicting protein functions and activities. Overall, the consistent improvements verify the benefits of injecting functional information into sequence-structure models, as done in ProtGO's teacher-student framework. The results cement ProtGO's effectiveness using knowledge distillation techniques.

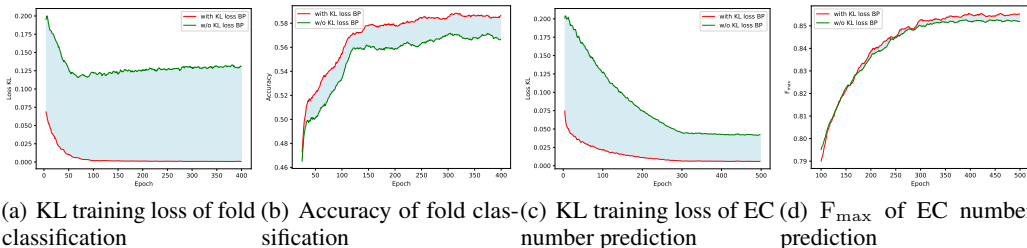

(a) KL training loss of fold classification (b) Accuracy of fold classification (c) KL training loss of EC number prediction (d) $F_{max}$ of EC number prediction

Figure 3: The KL training loss curves (a), (c) and test performance (b), (d) on the tasks of fold classification and EC number prediction. The red curve denotes that $\mathcal{L}_{kd}$ conducts its function, while the green curve denotes we calculated the value of $\mathcal{L}_{kd}$, but it is not involved in the process of the gradient backpropagation (BP).

## 4.5 Ablation Study

Table 3 presents ablation studies of the proposed ProtGO model on the four downstream tasks. We examine the impact of removing the teacher model, which means removing the $\mathcal{L}_{kd}$. We also remove the annotation encoder in the teacher, which means that we incorporate function information into the loss function for the teacher models. As shown in Table 3, removing the teacher model altogether

Table 3: Ablation experiments of our proposed method. w/o AE-T denotes without the annotation encoder in the teacher model. w/o teacher means without the teacher model and directly using the student model, which also means without $\mathcal{L}_{kd}$.

| Method | Fold Classification | | | Enzyme | GO | | | EC |
| | Fold | Superfamily | Family | Reaction | BP | MF | CC | |
| --- | --- | --- | --- | --- | --- | --- | --- | --- |
| ProtGO | 60.5 | 79.4 | 99.8 | 89.4 | 0.464 | 0.667 | 0.492 | 0.857 |
| w/o AE-T | 60.4 | 79.1 | 99.7 | 88.9 | 0.454 | 0.664 | 0.490 | 0.854 |
| w/o Teacher | 57.8 | 78.7 | 99.6 | 88.6 | 0.458 | 0.660 | 0.484 | 0.851 |

(w/o Teacher) leads to substantial performance drops across all tasks compared to the full ProtGO. This shows the teacher's knowledge distillation provides useful signals for the student model. Besides, removing the annotation encoder in the teacher (w/o AE-T) also degrades performance, though less severely. Despite being a label-augmented strategy, the annotation encoder exhibits minimal influence, indicating low sensitivity and limited impact on test performance. Our student model is specifically designed to process protein sequences and structures as inputs, enabling it to function independently without the need for guidance from the teacher model.

Figure 3 illustrates the comparisons of the knowledge distillation loss $\mathcal{L}_{kd}$ with and without involvement in backpropagation during training. When $\mathcal{L}_{kd}$ is excluded from the gradient backpropagation process, it exhibits a decrease alongside the classification loss $\mathcal{L}_{\text{student}}$. However, its value remains substantially higher compared to when $\mathcal{L}_{kd}$ is included in the training process. Similar observations are presented for the accuracy and $F_{\text{max}}$ on the fold classification and EC number prediction. The notable disparity observed between the distillation loss and the test performance with and without involvement in backpropagation suggests that the KL loss does indeed play a significant role in guiding the student model's learning process. Its presence influences the model's performance and convergence.

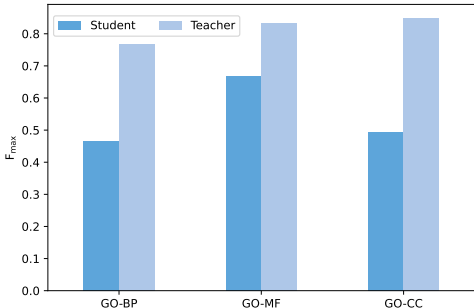

Figure 4: Performance comparisons of the teacher and student of ProtGO on GO term prediction.

We compare the performance of the teacher and the student on the tasks of GO term prediction. From the provided Figure 4, it is evident that incorporating functional information as the input of the annotation encoder significantly enhances performance, particularly for MF and CC term prediction. These two classes have fewer categories and are more accessible, resulting in higher scores.

## 5 Conclusion

In this paper, we proposed ProtGO, a multimodal protein representation learning framework integrating information from protein sequences, structures, and function annotations. While the teacher network requires extra information about function annotations as input, such is not always available for the student model. The student model amends the problem by mimicking the behavior or predictions of the teacher model. Our main focus is to obtain comprehensive embeddings for the student model, whereas the complete training of the teacher model is not our primary concern. We estimate the latent embedding distributions for the teacher-student model and learn annotation-enriched student representations by distribution approximation. Compared to the mainstream protein representation learning techniques, ProtGO achieves superior performances in predicting protein families, reactions, GO terms, and EC numbers. These consistent improvements across benchmarks highlight the advantages of this approach for informative protein representation learning. A limitation is that this base model is not pre-trained on large-scale datasets. One way for improvement is to integrate ProtGO with other pre-training strategies (refer to Appendix G). This would need to manage the computational resources required for training and inference and ensure compatibility between different model architectures and training objectives.

## Acknowledgements

This work was supported by the National Science and Technology Major Project (No. 2022ZD0115101), the National Natural Science Foundation of China Project (No. U21A20427 and No. 623B2086), Project (No. WU2022A009) from the Center of Synthetic Biology and Integrated Bioengineering of Westlake University and Project (No. WU2023C019) from the Westlake University Industries of the Future Research Funding.

## Footnotes

*Equal contribution. †Correspondence: {stan.zq.li}@westlake.edu.cn

[1]`https://geneontology.org/docs/ontology-documentation/`

## References

[1] Bozhen Hu, Jun Xia, Jiangbin Zheng, Cheng Tan, Yufei Huang, Yongjie Xu, and Stan Z Li. Protein language models and structure prediction: Connection and progression. *arXiv preprint arXiv:2211.16742*, 2022.

[2] John Jumper, Richard Evans, Alexander Pritzel, Tim Green, Michael Figurnov, Olaf Ronneberger, Kathryn Tunyasuvunakool, Russ Bates, Augustin Žídek, Anna Potapenko, et al. Highly accurate protein structure prediction with alphafold. *Nature*, 596(7873):583–589, 2021.

[3] Josh Abramson, Jonas Adler, Jack Dunger, Richard Evans, Tim Green, Alexander Pritzel, Olaf Ronneberger, Lindsay Willmore, Andrew J Ballard, Joshua Bambrick, et al. Accurate structure prediction of biomolecular interactions with alphafold 3. *Nature*, pages 1–3, 2024.

[4] Onur Serçinoğlu and Pemra Ozbek. Sequence-structure-function relationships in class i mhc: A local frustration perspective. *PloS one*, 15(5):e0232849, 2020.

[5] Serbulent Unsal, Heval Atas, Muammer Albayrak, Kemal Turhan, Aybar C Acar, and Tunca Doğan. Learning functional properties of proteins with language models. *Nature Machine Intelligence*, 4(3):227–245, 2022.

[6] Zeming Lin et al. Language models of protein sequences at the scale of evolution enable accurate structure prediction. *BioRxiv*, 2022:500902, 2022.

[7] Ahmed Elnaggar, Michael Heinzinger, Christian Dallago, Ghalia Rehawi, Yu Wang, Llion Jones, Tom Gibbs, Tamas Feher, Christoph Angerer, Martin Steinegger, et al. Prottrans: Toward understanding the language of life through self-supervised learning. *IEEE transactions on pattern analysis and machine intelligence*, 44(10):7112–7127, 2021.

[8] Alexander Rives et al. Biological structure and function emerge from scaling unsupervised learning to 250 million protein sequences. *Proceedings of the National Academy of Sciences*, 118(15):e2016239118, 2021.

[9] Roshan M Rao, Jason Liu, Robert Verkuil, Joshua Meier, John Canny, Pieter Abbeel, Tom Sercu, and Alexander Rives. Msa transformer. In *International Conference on Machine Learning*, pages 8844–8856. PMLR, 2021.

[10] Roshan M Rao, Joshua Meier, Tom Sercu, Sergey Ovchinnikov, and Alexander Rives. Transformer protein language models are unsupervised structure learners. *bioRxiv*, 2020. doi: 10.1101/2020.12.15.422761. URL https://www.biorxiv.org/content/10.1101/2020.12.15.422761v1.

[11] Bo Chen, Xingyi Cheng, Pan Li, Yangli-ao Geng, Jing Gong, Shen Li, Zhilei Bei, Xu Tan, Boyan Wang, Xin Zeng, et al. xtrimopglm: unified 100b-scale pre-trained transformer for deciphering the language of protein. *arXiv preprint arXiv:2401.06199*, 2024.

[12] Hehe Fan, Zhangyang Wang, Yi Yang, and Mohan Kankanhalli. Continuous-discrete convolution for geometry-sequence modeling in proteins. In *The Eleventh International Conference on Learning Representations*, 2023.

[13] Yue Liu, Ke Liang, Jun Xia, Sihang Zhou, Xihong Yang, , Xinwang Liu, and Z. Stan Li. Dink-net: Neural clustering on large graphs. In *Proc. of ICML*, 2023.

[14] Lirong Wu, Haitao Lin, Bozhen Hu, Cheng Tan, Zhangyang Gao, Zicheng Liu, and Stan Z Li. Beyond homophily and homogeneity assumption: Relation-based frequency adaptive graph neural networks. *IEEE Transactions on Neural Networks and Learning Systems*, 2023.

[15] Federico Baldassarre, David Menéndez Hurtado, Arne Elofsson, and Hossein Azizpour. Graphqa: protein model quality assessment using graph convolutional networks. *Bioinformatics*, page 360–366, Apr 2021. doi: 10.1093/bioinformatics/btaa714. URL `http://dx.doi.org/10.1093/bioinformatics/btaa714`.

[16] Pedro Hermosilla and Timo Ropinski. Contrastive representation learning for 3d protein structures. *arXiv preprint arXiv:2205.15675*, 2022.

[17] Zuobai Zhang, Minghao Xu, Arian Jamasb, Vijil Chenthamarakshan, Aurelie Lozano, Payel Das, and Jian Tang. Protein representation learning by geometric structure pretraining. In *International Conference on Learning Representations*, 2023.

[18] Fan Hu, Yishen Hu, Weihong Zhang, Huazhen Huang, Yi Pan, and Peng Yin. A multimodal protein representation framework for quantifying transferability across biochemical downstream tasks. *Advanced Science*, page 2301223, 2023.

[19] Hong-Yu Zhou, Yunxiang Fu, Zhicheng Zhang, Cheng Bian, and Yizhou Yu. Protein representation learning via knowledge enhanced primary structure modeling. *arXiv preprint arXiv:2301.13154*, 2023.

[20] Minghao Xu, Xinyu Yuan, Santiago Miret, and Jian Tang. Protst: Multi-modality learning of protein sequences and biomedical texts. In *International Conference on Machine Learning*, pages 38749–38767. PMLR, 2023.

[21] Michael Ashburner, Catherine A. Ball, Judith A. Blake, David Botstein, Heather Butler, J. Michael Cherry, Allan P. Davis, Kara Dolinski, Selina S. Dwight, Janan T. Eppig, Midori A. Harris, David P. Hill, Laurie Issel-Tarver, Andrew Kasarskis, Suzanna Lewis, John C. Matese, Joel E. Richardson, Martin Ringwald, Gerald M. Rubin, and Gavin Sherlock. Gene ontology: tool for the unification of biology. *Nature Genetics*, page 25–29, May 2000. doi: 10.1038/75556. URL `http://dx.doi.org/10.1038/75556`.

[22] UniProt Consortium. Update on activities at the universal protein resource (uniprot) in 2013. *Nucleic acids research*, 41(D1):D43–D47, 2012.

[23] Helen M. Berman, John D. Westbrook, Zukang Feng, Gary L. Gilliland, Talapady N. Bhat, Helge Weissig, Ilya N. Shindyalov, and Philip E. Bourne. The protein data bank. *Nucleic Acids Research*, 2000.

[24] Ningyu Zhang, Zhen Bi, Xiaozhuan Liang, Siyuan Cheng, Haosen Hong, Shumin Deng, Jiazhang Lian, Qiang Zhang, and Huajun Chen. Ontoprotein: Protein pretraining with gene ontology embedding. *arXiv preprint arXiv:2201.11147*, 2022.

[25] Mateo Torres, Haixuan Yang, Alfonso E Romero, and Alberto Paccanaro. Protein function prediction for newly sequenced organisms. *Nature Machine Intelligence*, 3(12):1050–1060, 2021.

[26] Nabil Ibtehaz, Yuki Kagaya, and Daisuke Kihara. Domain-pfp allows protein function prediction using function-aware domain embedding representations. *Communications Biology*, 6(1):1103, 2023.

[27] Roshan Rao, Nicholas Bhattacharya, Neil Thomas, Yan Duan, Peter Chen, John Canny, Pieter Abbeel, and Yun Song. Evaluating protein transfer learning with tape. *Advances in neural information processing systems*, 32, 2019.

[28] Erik Nijkamp, Jeffrey A Ruffolo, Eli N Weinstein, Nikhil Naik, and Ali Madani. Progen2: exploring the boundaries of protein language models. *Cell systems*, 14(11):968–978, 2023.

[29] Noelia Ferruz, Steffen Schmidt, and Birte Höcker. Protgpt2 is a deep unsupervised language model for protein design. *Nature Communications*, Jul 2022. doi: 10.1038/s41467-022-32007-7. URL `http://dx.doi.org/10.1038/s41467-022-32007-7`.

[30] Michael Heinzinger, Maria Littmann, Ian Sillitoe, Nicola Bordin, Christine Orengo, and Burkhard Rost. Contrastive learning on protein embeddings enlightens midnight zone. *NAR genomics and bioinformatics*, 4(2):lqac043, 2022.

[31] Pedro Hermosilla, Marco Schäfer, Matěj Lang, Gloria Fackelmann, Pere Pau Vázquez, Barbora Kozlíková, Michael Krone, Tobias Ritschel, and Timo Ropinski. Intrinsic-extrinsic convolution and pooling for learning on 3d protein structures. *International Conference on Learning Representations*, 2021.

[32] Limei Wang, Haoran Liu, Yi Liu, Jerry Kurtin, and Shuiwang Ji. Learning hierarchical protein representations via complete 3d graph networks. In *The Eleventh International Conference on Learning Representations*, 2023.

[33] Bowen Jing, Stephan Eismann, Patricia Suriana, Raphael JL Townshend, and Ron Dror. Learning from protein structure with geometric vector perceptrons. *arXiv preprint arXiv:2009.01411*, 2020.

[34] Yuzhi Guo, Jiaxiang Wu, Hehuan Ma, and Junzhou Huang. Self-supervised pre-training for protein embeddings using tertiary structures. In *Proceedings of the AAAI Conference on Artificial Intelligence*, volume 36, pages 6801–6809, 2022.

[35] Jacob Devlin, Ming-Wei Chang, Kenton Lee, and Kristina Toutanova. Bert: Pre-training of deep bidirectional transformers for language understanding. *arXiv preprint arXiv:1810.04805*, 2018.

[36] Geoffrey Hinton, Oriol Vinyals, and Jeff Dean. Distilling the knowledge in a neural network. *arXiv preprint arXiv:1503.02531*, 2015.

[37] Ke Liang, Lingyuan Meng, Meng Liu, Yue Liu, Wenxuan Tu, Siwei Wang, Sihang Zhou, Xinwang Liu, Fuchun Sun, and Kunlun He. A survey of knowledge graph reasoning on graph types: Static, dynamic, and multi-modal. *IEEE Transactions on Pattern Analysis and Machine Intelligence*, 2024.

[38] Jing Liu, Tongya Zheng, Guanzheng Zhang, and Qinfen Hao. Graph-based knowledge distillation: A survey and experimental evaluation. *arXiv preprint arXiv:2302.14643*, 2023.

[39] Yijun Tian, Chuxu Zhang, Zhichun Guo, Xiangliang Zhang, and Nitesh V Chawla. Nosmog: Learning noise-robust and structure-aware mlps on graphs. *arXiv preprint arXiv:2208.10010*, 2022.

[40] Wentao Zhang, Xupeng Miao, Yingxia Shao, Jiawei Jiang, Lei Chen, Olivier Ruas, and Bin Cui. Reliable data distillation on graph convolutional network. In *Proceedings of the 2020 ACM SIGMOD International Conference on Management of Data*, pages 1399–1414, 2020.

[41] Huarui He, Jie Wang, Zhanqiu Zhang, and Feng Wu. Compressing deep graph neural networks via adversarial knowledge distillation. In *Proceedings of the 28th ACM SIGKDD Conference on Knowledge Discovery and Data Mining*, pages 534–544, 2022.

[42] Ruifei He, Shuyang Sun, Jihan Yang, Song Bai, and Xiaojuan Qi. Knowledge distillation as efficient pre-training: Faster convergence, higher data-efficiency, and better transferability. In *Proceedings of the IEEE/CVF Conference on Computer Vision and Pattern Recognition*, pages 9161–9171, 2022.

[43] Abolfazl Farahani, Sahar Voghoei, Khaled Rasheed, and Hamid R Arabnia. A brief review of domain adaptation. *Advances in data science and information engineering: proceedings from ICDATA 2020 and IKE 2020*, pages 877–894, 2021.

[44] Garrett Wilson and Diane J Cook. A survey of unsupervised deep domain adaptation. *ACM Transactions on Intelligent Systems and Technology (TIST)*, 11(5):1–46, 2020.

[45] Mei Wang and Weihong Deng. Deep visual domain adaptation: A survey. *Neurocomputing*, 312:135–153, 2018.

[46] A Tuan Nguyen, Toan Tran, Yarin Gal, Philip HS Torr, and Atılım Güneş Baydin. Kl guided domain adaptation. *arXiv preprint arXiv:2106.07780*, 2021.

[47] Mingsheng Long, Yue Cao, Jianmin Wang, and Michael Jordan. Learning transferable features with deep adaptation networks. In *International conference on machine learning*, pages 97–105. PMLR, 2015.

[48] Yaroslav Ganin and Victor Lempitsky. Unsupervised domain adaptation by backpropagation. In *International conference on machine learning*, pages 1180–1189. PMLR, 2015.

[49] Mingsheng Long, Zhangjie Cao, Jianmin Wang, and Michael I Jordan. Conditional adversarial domain adaptation. *Advances in neural information processing systems*, 31, 2018.

[50] Zhongyi Pei, Zhangjie Cao, Mingsheng Long, and Jianmin Wang. Multi-adversarial domain adaptation. In *Proceedings of the AAAI conference on artificial intelligence*, volume 32, 2018.

[51] Kuniaki Saito, Donghyun Kim, Stan Sclaroff, Trevor Darrell, and Kate Saenko. Semi-supervised domain adaptation via minimax entropy. In *Proceedings of the IEEE/CVF international conference on computer vision*, pages 8050–8058, 2019.

[52] Taekyung Kim and Changick Kim. Attract, perturb, and explore: Learning a feature alignment network for semi-supervised domain adaptation. In *Computer Vision–ECCV 2020: 16th European Conference, Glasgow, UK, August 23–28, 2020, Proceedings, Part XIV 16*, pages 591–607. Springer, 2020.

[53] Pin Jiang, Aming Wu, Yahong Han, Yunfeng Shao, Meiyu Qi, and Bingshuai Li. Bidirectional adversarial training for semi-supervised domain adaptation. In *IJCAI*, pages 934–940, 2020.

[54] Can Qin, Lichen Wang, Qianqian Ma, Yu Yin, Huan Wang, and Yun Fu. Contradictory structure learning for semi-supervised domain adaptation. In *Proceedings of the 2021 SIAM International Conference on Data Mining (SDM)*, pages 576–584. SIAM, 2021.

[55] John Ingraham, Vikas Garg, Regina Barzilay, and Tommi Jaakkola. Generative models for graph-based protein design. *Advances in neural information processing systems*, 32, 2019.

[56] Samy Bengio, Jason Weston, and David Grangier. Label embedding trees for large multi-class tasks. *Advances in neural information processing systems*, 23, 2010.

[57] Xu Sun, Bingzhen Wei, Xuancheng Ren, and Shuming Ma. Label embedding network: Learning label representation for soft training of deep networks. *arXiv preprint arXiv:1710.10393*, 2017.

[58] Gang Xu, Qinghua Wang, and Jianpeng Ma. Opus-rota4: a gradient-based protein side-chain modeling framework assisted by deep learning-based predictors. *Briefings in Bioinformatics*, 23(1):bbab529, 2022.

[59] Jack Hanson, Kuldip Paliwal, Thomas Litfin, Yuedong Yang, and Yaoqi Zhou. Improving prediction of protein secondary structure, backbone angles, solvent accessibility and contact numbers by using predicted contact maps and an ensemble of recurrent and residual convolutional neural networks. *Bioinformatics*, 35(14):2403–2410, 2019.

[60] Diederik P Kingma and Max Welling. Auto-encoding variational bayes. *arXiv preprint arXiv:1312.6114*, 2013.

[61] Amir Shanehsazzadeh, David Belanger, and David Dohan. Is transfer learning necessary for protein landscape prediction? *arXiv preprint arXiv:2011.03443*, 2020.

[62] Thomas N Kipf and Max Welling. Semi-supervised classification with graph convolutional networks. In *ICLR*, 2017.

[63] Petar Velickovic, Guillem Cucurull, Arantxa Casanova, Adriana Romero, Pietro Lio, Yoshua Bengio, et al. Graph attention networks. In *ICLR*, 2018.

[64] Georgy Derevyanko, Sergei Grudinin, Yoshua Bengio, and Guillaume Lamoureux. Deep convolutional networks for quality assessment of protein folds. *Bioinformatics*, 34(23):4046–4053, 2018.

[65] Federico Baldassarre, David Menéndez Hurtado, Arne Elofsson, and Hossein Azizpour. Graphqa: protein model quality assessment using graph convolutional networks. *Bioinformatics*, 2020.

[66] Bowen Jing, Stephan Eismann, Patricia Suriana, Raphael JL Townshend, and Ron Dror. Learning from protein structure with geometric vector perceptrons. *arXiv preprint arXiv:2009.01411*, 2020.

[67] Vladimir Gligorijević, P Douglas Renfrew, Tomasz Kosciolek, Julia Koehler Leman, Daniel Berenberg, Tommi Vatanen, Chris Chandler, Bryn C Taylor, Ian M Fisk, Hera Vlamakis, et al. Structure-based protein function prediction using graph convolutional networks. *Nature communications*, 12(1):3168, 2021.

[68] Jie Hou, Badri Adhikari, and Jianlin Cheng. Deepsf: deep convolutional neural network for mapping protein sequences to folds. *Bioinformatics*, 34(8):1295–1303, 2018.

[69] Edwin C Webb et al. *Enzyme nomenclature 1992. Recommendations of the Nomenclature Committee of the International Union of Biochemistry and Molecular Biology on the Nomenclature and Classification of Enzymes.* Number Ed. 6. Academic Press, 1992.

[70] Marina V Omelchenko, Michael Y Galperin, Yuri I Wolf, and Eugene V Koonin. Non-homologous isofunctional enzymes: a systematic analysis of alternative solutions in enzyme evolution. *Biology direct*, 5:1–20, 2010.

[71] Helen M Berman, John Westbrook, Zukang Feng, Gary Gilliland, Talapady N Bhat, Helge Weissig, Ilya N Shindyalov, and Philip E Bourne. The protein data bank. *Nucleic acids research*, 28(1):235–242, 2000.

[72] UniProt Consortium. Uniprot: a worldwide hub of protein knowledge. *Nucleic acids research*, 47(D1):D506–D515, 2019.

[73] Bozhen Hu, Cheng Tan, Jun Xia, Jiangbin Zheng, Yufei Huang, Lirong Wu, Yue Liu, Yongjie Xu, and Stan Z Li. Learning complete protein representation by deep coupling of sequence and structure. *bioRxiv*, pages 2023–07, 2023.

[74] Microsoft. Neural Network Intelligence, 1 2021. URL `https://github.com/microsoft/nni`.

[75] Maxat Kulmanov et al. Protein function prediction as approximate semantic entailment. *Nature Machine Intelligence*, 6(2):220–228, 2024.

[76] Youhan Lee, Hasun Yu, Jaemyung Lee, and Jaehoon Kim. Pre-training sequence, structure, and surface features for comprehensive protein representation learning. In *ICLR*, 2023.

[77] Z Zhang, C Wang, M Xu, V Chenthamarakshan, AC Lozano, P Das, and J Tang. A systematic study of joint representation learning on protein sequences and structures. *Preprint at http://arxiv.org/abs/2303.06275*, 2023.

# A  Social Impact

Our proposed framework, ProtGO, can enable advanced protein analyses and provide effective and comprehensive representations that incorporate the information from protein sequences, structures, and functions. However, there may exist broader impacts and harmful activities. In detail, the representations are potentially misused, e.g., designing harmful molecules or proteins based on these representations. Wet lab experiments may be needed for the newly found mechanisms or functions of proteins based on the learned representations.

# B  Task

**Fold Classification.**   In order to understand how protein structure and evolution interact, it is crucial to be able to predict fold classes [68]. This dataset contains 16,712 total proteins across 1,195 fold classes. Three test sets are provided. Fold: proteins from the same superfamily are excluded during training; SuperFamily: proteins from the same family are not used for training; and Family: the training set includes proteins from the same family.

**Enzyme Reaction Classification.**   Enzyme reaction classification can be viewed as a protein function prediction task based on the enzyme-catalyzed reactions defined by the four levels of enzyme commission numbers [69, 70]. We use the dataset [31, 71] containing 29,215 training proteins, 2,562 validation proteins, and 5,651 test proteins, spanning 384 four-level EC classes.

**GO Term Prediction.**   The aim of Gene Ontology (GO) term prediction is to predict whether a given protein should be annotated with a particular GO term. As we have stated before, proteins are categorized into three hierarchical ontologies: molecular function (MF), biological process (BP), and cellular component (CC). Specifically, MF denotes molecular activities of a protein, BP refers to larger biological processes it is involved in, and CC describes subcellular locations and extracellular components [72]. Accurately assigning GO terms is crucial for understanding protein function and assessing computational methods.

**EC Number Prediction.**   This task aims to predict the 538 Enzyme Commission (EC) numbers at the third and fourth level hierarchies for different proteins [67], which provide precise information about a protein's enzymatic function, based on the protein's features. The large number of classes at the third and fourth EC levels makes this a challenging multi-class prediction problem in bioinformatics.

# C  Evaluation Metric

$F_{\max}$ provides an overall metric that combines both accuracy and coverage of the predictions. It is calculated by first determining the precision and recall for each protein, then averaging these results over all proteins [17, 67]. $p_i^j$ is the prediction probability for the $j$-th class of the $i$-th protein, given the decision threshold $t \in [0, 1]$, the precision and call are given as:

$$\text{precision}_i(t) = \frac{\sum_j \mathbb{I}[((p_i^j \geq t) \cap b_i^j)]}{\sum_j \mathbb{I}[(p_i^j \geq t)]}, \quad \text{recall}_i(t) = \frac{\sum_j \mathbb{I}[((p_i^j \geq t) \cap b_i^j)]}{\sum_j b_i^j}$$

where $b_i^j \in \{0, 1\}$ is the corresponding binary class label, and $\mathbb{I} \in \{0, 1\}$ is an indicator function. If there are $N$ proteins in total, these protein-level precision and recall values are averaged over all proteins to obtain the overall precision and recall for the dataset, then the average precision and recall are defined as:

$$\text{precision}(t) = \frac{\sum_i^N \text{precision}_i(t)}{\sum_i^N \left( \left( \sum_j (p_i^j \geq t) \right) \geq 1 \right)}, \quad \text{recall}(t) = \frac{\sum_i^N \text{recall}_i(t)}{N}$$

Finally, $F_{\max}$ is defined as the maximum value of the F-score over all thresholds.

$$F_{\max} = \max_t \left\{ \frac{2 \cdot \text{precision}(t) \cdot \text{recall}(t)}{\text{precision}(t) + \text{recall}(t)} \right\} \tag{12}$$

## D   Complexity Analysis

Our main focus is on generating comprehensive embeddings for the student model, with less emphasis placed on the training specifics of the teacher model. Regarding the student model, the computational complexity of one message passing layer in this framework is $\mathcal{O}(nd_n)$, where $d_n$ represents the average node degree, typically much smaller than $n$. The time complexity is directly related to the computational complexity of the message passing layer; as graph convolution is performed on nodes and edges simultaneously, the time complexity remains $\mathcal{O}(nd_n)$, linear with the number of residues $n$. Denoting the size of the batch as $B_s$, the overall computational complexity is merely $\mathcal{O}(B_s nd_n)$.

## E   Experiment Setup

### E.1   Model Details

The radius $r_s$ threshold increases from 4 to 16, and $l_s$ is 11. We set two message passing layers with one average sequence pooling per GNN. After the pooling layer, the number of residues is halved, and we update the edge conditions before performing another round of message passing and pooling operations, as illustrated in the model Figure 2. The final GNNs include eight message-massing and four pooling layers, which are sufficient for achieving satisfactory results. The number of initial feature channels is 256, increased to 2048. The annotation encoder has 2 FC layers changing feature channels from 2752 to 2048. The classification head is a liner layer for predicting classes. For the teacher model, we use $z_S$ to get the predicted annotations by the classification head and calculate the loss by $\mathcal{L}_{\text{teacher}}$. The final loss $\mathcal{L}$ is used for the training of the student model.

We know spatially adjacent residues can still exist even when the sequence distance is large [73]. We perform sequence average pooling and change edge conditions after once pooling. These operations enable the protein graph to cover more distant nodes.

### E.2   Training Details

Dataset statistics [17] of the four downstream tasks are summarized in Table 4. The proposed framework conducted experiments on NVIDIA-SMI A100 GPUs and NVIDIA Tesla V100 GPUs, implemented with PyTorch 1.13+cu117 and PyTorch Geometric 2.3.1 with CUDA 11.2.

Table 4: Dataset statistics. #X means the number of X.

| Dataset | #Train | #Validation | #Test |
|---|---|---|---|
| Enzyme Commission | $15,550$ | $1,729$ | $1,919$ |
| Gene Ontology | $29,898$ | $3,322$ | $3,415$ |
| Fold Classification - Fold | $12,312$ | $736$ | $718$ |
| Fold Classification - Superfamily | $12,312$ | $736$ | $1,254$ |
| Fold Classification - Family | $12,312$ | $736$ | $1,272$ |
| Reaction Classification | $29,215$ | $2,562$ | $5,651$ |

In biology, a linear combination of original data with Gaussian noise [34] is a simple but effective way to augment the protein data:

$$(P_i, \boldsymbol{x}_i) \leftarrow (P_i, \boldsymbol{x}_i) + \Theta, \Theta \sim (\mu_k, \sigma_k^2) \tag{13}$$

where $\mu_k$ and $\sigma_k$ are selected as the random noise's mean (expectation) and standard deviation.

Hyper-parameters related to the networks are set the same across different datasets: Adam optimizer with learning rate $l_r = 1e-3$, weight decay $decay = 5e-4$, epochs $T = 300$, Gaussian noise $\mu_k = 0, \sigma_k = 0.1$, it indicates trivial perturbation is introduced to the protein native structures.

The other dataset-specific hyper-parameters are determined by an AutoML toolkit NNI [74] with the search spaces. The loss weight hyper-parameter is related to the value of the task-specific loss $\beta = \{1, 0.1, 0.01, 0.001, 0\}$, and $\alpha = \{10, 1, 0.1, 0.01, 0.001, 0\}$. As for the batch size and training epochs, etc., which influence the convergence speed of deep learning models, we set 16 and 500 respectively.

# F  KL Guided Domain Adaption

Assuming source and target domains have the same support set and share the representation mapping $p(z|G)$, this means these two domains have the same datasets of protein graphs and functions. Given the representation $z$, we learn a classifier to predict the label $y$ through the predictive distribution $\hat{p}(y|z)$ that is an approximation of the ground truth. During training, the representation network $p(z|G)$ and the classifier $\hat{p}(y|z)$ are trained jointly on the source domain and we hope that they can generalize to the target domain, meaning that both $p(z|G)$ and $\hat{p}(y|z)$ are kept unchanged between training and testing.

We define the predictive distribution of $y$ given $G$ as

$$\hat{p}(y|G) = \mathbb{E}_{p(z|G)}[\hat{p}(y|z)] \tag{14}$$

We have a single $z$ from the source model $p(z|G)$ for each protein. The training objective of the source domain is

$$\mathcal{L}_{\text{teacher}} = \mathbb{E}_{G,y \sim p_S(G,y), z \sim p(z|G)}[-\log \hat{p}(y|z)] = \mathbb{E}_{p_S(z,y)}[-\log \hat{p}(y|z)] \tag{15}$$

We consider the two assumptions of the representation $z$ on the source domain:

**Assumption 1.** $I_S(z,y) = I_S(G,y)$, where $I_S(\cdot,\cdot)$ is the mutual information term, calculated on the source domain. In particular:

$$I_S(z,y) = \mathbb{E}_{p_S(z,y)}\left[\log \frac{p_S(z,y)}{p_S(z)p_S(y)}\right]; \quad I_S(G,y) = \mathbb{E}_{p_S(G,y)}\left[\log \frac{p_S(G,y)}{p_S(G)p_S(y)}\right] \tag{16}$$

The mutual information quantifies the amount of information shared between the variables $z$ and $y$ (or $G$ and $y$) in the source domain. It measures the dependence or correlation between these variables in the context of the source domain data. This is often referred to as the 'sufficiency assumption' since it indicates that the representation $z$ has the same information about the label $y$ as the original input protein graph $G$, and is sufficient for this prediction task in the source domain. Note that the data processing inequality indicates that $I_S(z,y) \leq I_S(G,y)$, so here we assume that $z$ contains maximum information about $y$.

**Assumption 2.** $p_S(y|G) = \mathbb{E}_{p(z|G)}[p_S(y|z)]$

When this assumption holds, the predictive distribution $\hat{p}(y|G)$ will approximate $p_S(y|G)$, as long as $\hat{p}(y|z)$ approximates $p_S(y|z)$.

The above two assumptions ensure that the teacher network has good performance in the source domain. Now, we continue to consider the test loss and how we can reduce it. The loss of the target domain is:

$$\begin{aligned}
\mathcal{L}_{\text{student}}^* &= \mathbb{E}_{p_T(G,y)}[-\log \hat{p}(y|G)] = \mathbb{E}_{p_T(G,y)}\left[-\log \mathbb{E}_{p(z|G)}[\hat{p}(y|z)]\right] \\
&\leq \mathbb{E}_{p_T(G,y)}\left[\mathbb{E}_{p(z|G)}[-\log \hat{p}(y|z)]\right] \\
&= \mathbb{E}_{p_T(z,y)}[-\log \hat{p}(y|z)]
\end{aligned} \tag{17}$$

Since we do not know the target domain and the target data distribution, there is no way to guarantee the invariance (both marginally and conditionally) of the representation $z$. Therefore, We introduce the following proposition that ensures a generalization bound of the target domain loss based on the source domain loss and the KL divergence:

**Proposition 1.** If the loss $-\log \hat{p}(y|z)$ is bounded by $M$, we have:

$$\begin{aligned}
\mathcal{L}_{\text{student}}^* &\leq \mathcal{L}_{\text{teacher}} + \frac{M}{\sqrt{2}}\sqrt{\text{KL}\left[p_S(y,z) \parallel p_T(y,z)\right]} \\
&= \mathcal{L}_{\text{teacher}} + \frac{M}{\sqrt{2}}\sqrt{\text{KL}\left[p_S(z) \parallel p_T(z)\right] + \mathbb{E}_{p_S(z)}\left[\text{KL}\left[p_S(y|z) \parallel p_T(y|z)\right]\right]}
\end{aligned} \tag{18}$$

**Proposition 2.** If Assumption 1 and 2 hold, and if $\frac{p_S(G,y)}{p_T(G,y)} < \infty$ (i.e., there exists $N'$, which can be arbitrarily large, such that $\frac{p_S(G,y)}{p_T(G,y)} < N'$), we have

$$\mathbb{E}_{p_S(G)}\left[\text{KL}\left[p_S(y|z) \parallel p_T(y|z)\right]\right] \leq \mathbb{E}_{p_S(G)}\left[\text{KL}\left[p_S(y|G) \parallel p_T(y|G)\right]\right] \tag{19}$$

Table 5: The comparison results with pre-training methods ($F_{max}$) on GO term and EC number prediction. The best results are shown in bold.

| Category | Method | GO-BP | GO-MF | GO-CC | EC |
|---|---|---|---|---|---|
| Sequence | ESM-1b [8] | 0.452 | 0.659 | 0.477 | 0.869 |
| | ESM-2 [6] | 0.472 | 0.662 | 0.472 | 0.874 |
| Sequence-Function | ProtST-ESM-1b [20] | 0.480 | 0.661 | 0.488 | 0.878 |
| | ProtST-ESM-2 [20] | 0.482 | 0.668 | 0.487 | 0.878 |
| | DeepGO-SE [75] | 0.438 | 0.564 | 0.427 | 0.810 |
| Sequence-Structure | ESM-GearNet [77] | 0.516 | 0.684 | 0.506 | 0.890 |
| | GearNet-ESM-INR-MC [76] | 0.518 | 0.683 | 0.504 | **0.896** |
| | ProtGO-ESM (Student) | **0.520** | **0.693** | **0.536** | 0.887 |

This shows that the conditional misalignment in the representation space is bounded by the conditional misalignment in the input space. It then follows that:

$$\mathcal{L}_{\text{student}}^* \leq \mathcal{L}_{\text{teacher}} + \frac{M}{\sqrt{2}}\sqrt{\text{KL}\left[p_S(z) \parallel p_T(z)\right] + \mathbb{E}_{p_S(G)}\left[\text{KL}\left[p_S(y|G) \parallel p_T(y|G)\right]\right]} \qquad (20)$$

We know $y$ can represent the underlying functional label for the student model. Although the student model may not have these functional labels, but we can assume that they exist for theoretical reasons. The derived misalignment Eq. 20 and the derived loss Eq. 8 are based on the assumption that the source and target domains have the same support set. Thus, the loss of Eq. 8 can be used in an unsupervised way for the student to predict functions. However, the student model is applied to different downstream tasks, like classification, which has classification classes. Thus, we add the supervised student loss $\mathcal{L}_{\text{student}}$ and the knowledge distillation loss the $\mathcal{L}_{kd}$ as the final hybrid loss for the student to improve its performance on classification tasks.

## G  Comparisons with Pre-training Methods

To ensure fair comparisons with pre-trained models, we integrate ESM-2 (650M) [6] into ProtGO, resulting in the creation of ProtGO-ESM. In this approach, ESM embeddings are utilized to enhance the graph node features. Our assessment evaluates ProtGO-ESM against various pre-training techniques in the tasks of protein function and EC number prediction. This evaluation includes a range of methods, such as sequence-based approaches like ESM-1b [8] and ESM-2 [6]; sequence-function models including ProtST [20] and DeepGO-SE [75]; and sequence-structure methodologies such as GearNet-ESM [17] and GearNet-ESM-INR-MC [76]. Detailed comparative results are presented in Table 5. Notably, our proposed model, ProtGO-ESM, outperforms other methods across these sequence-based, sequence-structure-based, and sequence-function-based pre-training techniques.

